# Analog VLSI Model of Intersegmental Coordination With Nearest-Neighbor Coupling

**Girish N. Patel**
girish@ece.gatech.edu

**Jeremy H. Holleman**
jeremy@ece.gatech.edu

**Stephen P. DeWeerth**
steved@ece.gatech.edu

School of Electrical and Computer Engineering
Georgia Institute of Technology
Atlanta, Ga. 30332-0250

## Abstract

We have a developed an analog VLSI system that models the coordination of neurobiological segmental oscillators. We have implemented and tested a system that consists of a chain of eleven pattern generating circuits that are synaptically coupled to their nearest neighbors. Each pattern generating circuit is implemented with two silicon Morris-Lecar neurons that are connected in a reciprocally inhibitory network. We discuss the mechanisms of oscillations in the two-cell network and explore system behavior based on isotropic and anisotropic coupling, and frequency gradients along the chain of oscillators.

## 1 INTRODUCTION

In recent years, neuroscientists and modelers have made great strides towards illuminating structure and computational properties in biological motor systems. For example, much progress has been made toward understanding the neural networks that elicit rhythmic motor behaviors, including leech heartbeat (Calabrese and De Schutter, 1992), crustacean stomatogastric mill (Selverston, 1989) and tritonia swimming (Getting, 1989). In particular, segmented locomotory systems, such as those that underlie swimming in the lamprey (Cohen and Kiemel, 1993, Sigvardt, 1993, Grillner et al, 1991) and in the leech (Friesen and Pearce, 1993), are interesting from an quantitative perspective. In these systems, it is clear that coordinated motor behaviors are a result of complex interactions among membrane, synaptic, circuit, and system properties. However, because of the lack of sufficient neural underpinnings, a complete understanding of the computational principles in these systems is still lacking. Abstracting the biophysical complexity by modeling segmented systems as coupled nonlinear oscillators is one approach that has provided much insight into the operation of these systems (Cohen et al, 1982). More specifically, this type of modeling work has illuminated computational properties that give rise to *phase constancy*, a motor behavior that is characterized by intersegmental phase lags that are maintained at constant values independent of swimming frequency. For example, it has been shown that frequency gradients and asymmetrical coupling play an important role in establishing phase lags of correct sign and amplitude (Kopell and Ermentrout, 1988) as well as appropriate boundary conditions (Williams and Sigvardt, 1994).

Although theoretical modeling has provided much insight into the operation of interseg-

mental systems, these models have limited capacity for incorporating biophysical properties and complex interconnectivity. Software and/or hardware emulation provides the potential to add such complexity to system models. Additionally, the modularity and regularity in the anatomical and computational structures of intersegmental systems facilitate scalable representations. These factors make segmented systems particularly viable for modeling using neuromorphic analog very large-scale integrated (aVLSI) technology. In general, biological motor systems have a number of properties that make their real-time modeling using aVLSI circuits interesting and approachable. Like their sensory counterparts, they exhibit rich emergent properties that are generated by collective architectures that are regular and modular. Additionally, the fact that motor processing is at the periphery of the nervous system makes the analysis of the system behavior accessible due to the fact that output of the system (embodied in the motor actions) is observable and facilitates functional analysis.

The goals in this research are i) to study how the properties of individual neurons in a network affect the overall system behavior; (ii) to facilitate the validation of the principles underlying intersegmental coordination; and (iii) to develop a real-time, low power, motion control system. We want to exploit these principles and architectures both to improve our understanding of the biology and to design artificial systems that perform autonomously in various environments. In this paper we present an analog VLSI model of intersegmental coordination that addresses the role of frequency gradients and asymmetrical coupling. Each segment in our system is implemented with two silicon model neurons that are connected in a reciprocally inhibitory network. A model of intersegmental coordination is implemented by connecting eleven such oscillators, with nearest neighbor coupling. We present the neuron model, and we investigate the role of frequency gradients and asymmetrical coupling in the establishment of phase lags along a chain these neural oscillators.

## 2 NEURON MODEL

In order to produce bursting activity, a neuron must possess "slow" intrinsic time constants in addition to the "fast" time constants that are necessary for the generation of spikes. Hardware models of neurons with both slow and fast time constants have been designed based upon previously described Hodgkin–Huxley neuron models (Mahowald and Douglas, 1991). Although these circuits are good models of their biological counterparts, they are relatively complex, with a large parameter space and transistor count, limiting their usefulness in the development of large-scale systems. It has been shown (Skinner, 1994), however, that pattern generation can be represented with only the slow time constants, creating a system that represents the envelope of the bursting oscillations without the individual spikes. Model neurons with only slow time constants have been proposed by Morris and Lecar (1981).

We have implemented an analog VLSI model of the Morris-Lecar Neuron (Patel and DeWeerth, 1997). Figure 1 shows the circuit diagram of this neuron. The model consists of two state variables: one corresponding to the membrane potential ($V$) and one corresponding to a slow variable ($N$). The slow variable is obtained by delaying the mem-

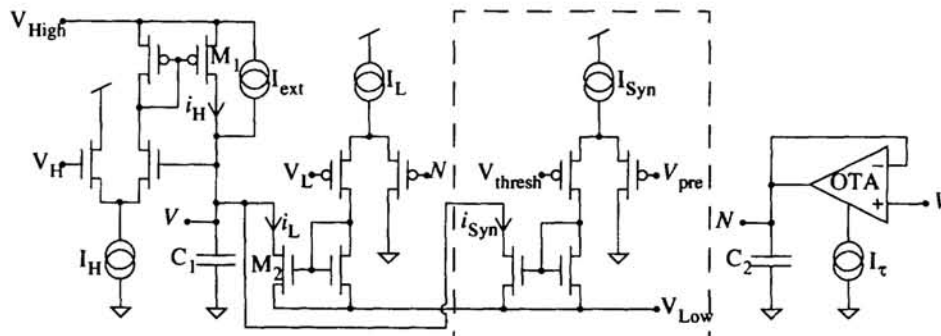

Figure 1: Circuit diagram of silicon Morris-Lecar Neuron

brane potential by way of an operational transconductance amplifier (OTA) connected in unity gain configuration with load capacitor $C_2$. The membrane potential is obtained by injecting two positive currents ($I_{ext}$ and $i_H$) and two negative currents ($i_L$ and $i_{syn}$) into capacitor $C_1$. Current $i_H$ raises the membrane potential towards $V_{High}$ when the membrane potential increases above $V_H$, whereas current $i_L$ lowers the membrane potential towards $V_{Low}$ when the delayed membrane potential increases above $V_L$. The synaptic current, $i_{syn}$, activates when the presynaptic input, $V_{Pre}$, increases above $V_{thresh}$. Assuming operation of transistors in weak inversion and synaptic coupling turned off ($i_{syn} = 0$) the equations of motion for the system are.

$$C_1 \dot{V} = I_1(V,N) = I_{ext}\alpha_P + I_H \frac{\exp(\kappa(V-V_H)/U_T)}{1+\exp(\kappa(V-V_H)/U_T)}\alpha_P - I_L\frac{\exp(\kappa(N-V_L)/U_T)}{1+\exp(\kappa(N-V_L)/U_T)}\alpha_N$$

$$C_2 \dot{N} = I_2(V,N) = I_\tau \tanh(\kappa(V-N)/(2U_T))(1-\exp((N-V_{dd})/U_T))$$

The terms $\alpha_P$ and $\alpha_N$, where $\alpha_P = 1 - \exp(V-V_{High})/U_T$ and $\alpha_N = 1 - \exp(V_{Low} - V)/U_T$, correspond to the ohmic effect of transistor M1 and M2 respectively. $\kappa$ corresponds to the back-gate effect of a MOS transistor operated in weak inversion, and $U_T$ corresponds to the thermal voltage. We can understand the behavior of this circuit by analyzing the geometry of the curves that yield zero motion (i.e., when $I_1(V,N) = I_2(V,N) = 0$). These curves, referred to as nullclines, are shown in Figure 2 for various values of external current.

The externally applied constant current ($I_{ext}$), which has the effect of shifting the $V$ nullcline in the positive vertical direction (see Figure 2), controls the mode of operation of the neuron. When the $V$- and $N$ nullclines intersect between the local minimum and local maximum of the $V$ nullcline (P2 in Figure 2), the resulting fixed point is unstable and the trajectories of the system approach a stable limit-cycle (an *endogenous bursting* mode). Fixed points to the left of the local minimum (P1 in Figure 2) or to the right of the local maximum (P3 in Figure 2) are stable and correspond to a *silent* mode and a *tonic* mode of the neuron respectively. An inhibitory synaptic current ($i_{syn}$) has the effect of shifting the $V$ nullcline in the negative vertical direction; depending on the state of a presynaptic cell, $i_{Syn}$ can dynamically change the mode of operation of the neuron.

## 3 TWO-CELL NETWORK

When two cells are connected in a reciprocally inhibitory network, the two cells will oscillate in antiphase depending on the conditions of the free and inhibited cells and the value of the synaptic threshold (Skinner et. al, 1994). We assume that the turn-on characteristics of the synaptic current is sharp (valid for large $V_{High} - V_{Low}$) such that when the membrane potential of a presynaptic cell reaches above $V_{thresh}$, the postsynaptic cell is immediately inhibited by application of negative current $I_{Syn}$ to its membrane potential.

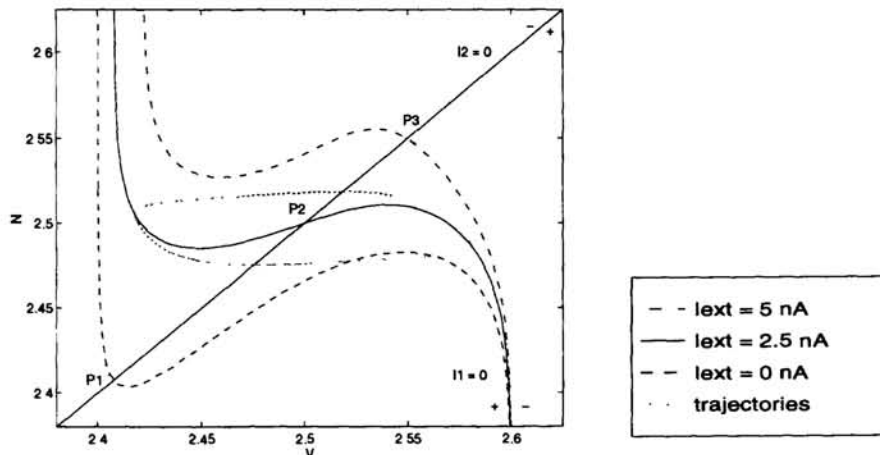

Figure 2: Nullcline and corresponding trajectories of silicon Morris-Lecar neuron.

If the free cell is an endogenous burster, the inhibited cell is silent, and the synaptic threshold is between the local maximum of the free cell and the local minimum in the inhibited cell, the mechanism for oscillation is due to *intrinsic release*. This mechanism can be understood by observing that the free cell undergoes rapid depolarization when its state approaches the local maximum thus facilitating the release of the inhibited cell. If the free cell is tonic and the inhibited cell is an endogenous burster (and conditions on synaptic threshold are the same as in the intrinsic release case), then the oscillations are due to an *intrinsic escape* mechanism. This mechanism is understood by observing that the inhibited cell undergoes rapid hyperpolarization, thus escaping inhibition, when its state approaches the local minimum. Note, in both intrinsic release and intrinsic escape mechanisms, the synaptic threshold has no effect on oscillator period because rapid changes in membrane potential occur before the effect of synaptic threshold.

When the free cell is an endogenous burster, the inhibited cell is silent, and the synaptic threshold is to the right of the local maximum of the free cell, then the oscillations are due to a *synaptic release* mechanism. This mechanism can be understood by observing that when the membrane potential of the free cell reaches below the synaptic threshold, the free cell ceases to inhibit the other cell which causes the release of the inhibited cell. When the free cell is tonic, and the inhibited cell is an endogenous burster, and the synaptic threshold is to the left of the local minimum of the inhibited cell, then the oscillations are due to a *synaptic escape* mechanism. This mechanism can be understood by observing that when the membrane potential of the inhibited cell crosses above the synaptic threshold, then the membrane potential of the inhibited cell is large enough to inhibit the free cell. Note, increasing the synaptic threshold has the effect of increasing oscillator frequency for the synaptic release mechanism, however, oscillator frequency under the synaptic escape mechanism will decrease with an increase in the synaptic threshold.

By setting $V_{High} - V_{Low}$ to a large value, the synaptic currents appear to have a sharp cutoff. However, because transistor currents saturate within a few thermal voltages, the nullclines due to the membrane potential appear less cubic-like and more square-like. This does not effect the qualitative behavior of the circuit, as we are able to produce antiphasic oscillations due to all four mechanisms. Figure 3 illustrates the four modes of oscillations under various parameter regimes. Figure 3A show typical waveforms from two silicon neurons when they are configured in a reciprocally inhibitory network. The oscillations in this case are due to intrinsic release mechanism and the frequency of oscillations are insensitive to the synaptic threshold. When the synaptic threshold is increased above 2.5 volts, the oscillations are due to the synaptic release mechanism and the oscillator frequency will increase as the synaptic threshold is increased, as shown in Figure 3C. By adjusting $I_{ext}$ such that the free cell is tonic and the inhibited cell bursts endogenously, we are able to produce oscillations due to the intrinsic escape mechanism, as

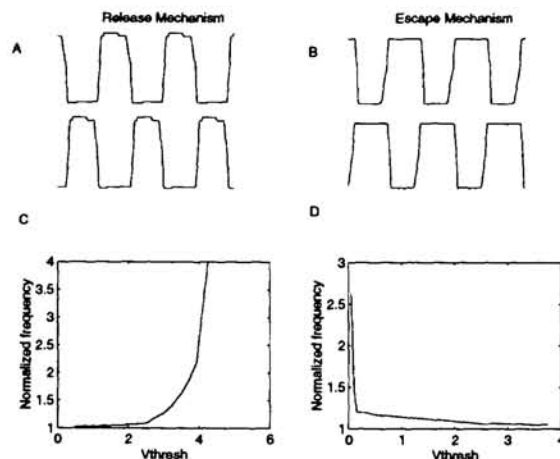

Figure 3: Experimental results from two neurons connected in a reciprocally inhibitory network. Antiphasic oscillations due to intrinsic release mechanism (A), and intrinsic escape mechanism (B). Dependence of oscillator frequency on synaptic threshold for the synaptic release mechanism (C) and synaptic escape mechanism (D)

shown in Figure 3B. As the synaptic threshold is decreased below 0.3 volts, the oscillations are caused by the synaptic escape mechanism and oscillator frequency increases as the synaptic threshold is decreased. The sharp transition between intrinsic and synaptic mechanisms is due to nullclines that appear square-like.

## 4 CHAIN OF COUPLED NEURAL OSCILLATORS

In order to build a chain of pattern generating circuits with nearest neighbor coupling, we designed our silicon neurons with five synaptic connections. The connections are made using the synaptic spread rule proposed by Williams (1990). The rule states that a neuron in any given segment can only connect to neurons in other segments that are homologues to the neurons it connects to in the local segment. Therefore, each neuron makes two inhibitory, contralateral connections and two excitatory, ipsilateral connections (as well a single inhibitory connection in the local segment). The synaptic circuit, shown in the dashed box in Figure 1, is repeated for each inhibitory synapse and its complementary version is repeated for the excitatory synapses. In order to investigate the role of frequency gradients, each neural oscillator has an independent parameter, $I_{ext}$, for setting the intrinsic oscillator period. A set of global parameters, $I_L$, $I_H$, $I_\tau$, $V_H$, $V_L$, $V_{High}$, and $V_{Low}$ control the mechanism of oscillation. These parameters are set such that the mechanism of oscillation is intrinsic release.

Because of inherent mismatch of devices in CMOS technology, a consequence in our model is that neurons with equal parameters do not necessarily behave with similar performance. Figure 4A illustrates the intrinsic oscillator period along the length of system when all neurons receive the same parameters. When the oscillators are symmetrically coupled, the resulting phase differences along the chain are nonzero, as shown in Figure 4B. The phase lags are negative with respect to the head position, thus the default swim direction is backward. As the coupling strength is increased, indicated by the lowermost curves in Figure 4B, the phase lags become smaller, as expected, but do not diminish to produce synchronous oscillations. When the oscillators are locked to one common frequency, $\Omega$, theory predicts (Kopell and Ermentrout, 1988) that the common frequency is dependent on intrinsic oscillator frequencies, and coupling from neighboring oscillators. In addition, under the condition of weak coupling, the effect of coupling can be quantified with coupling functions that depends on the phase difference between neighboring oscillators:

$$\Omega = \omega_i + H_A^i(\phi_i) + H_D^i(-\phi_{i-1})$$

where, $\omega_i$ is the intrinsic frequency of a given oscillator, $H_A$ and $H_D$ are coupling functions in the ascending and descending directions respectively, and $\phi_i$ is the phase difference between the (i+1)th and ith oscillator. This equation suggests that the phase lags must be large in order to compensate for large variations in the intrinsic oscillator fre-

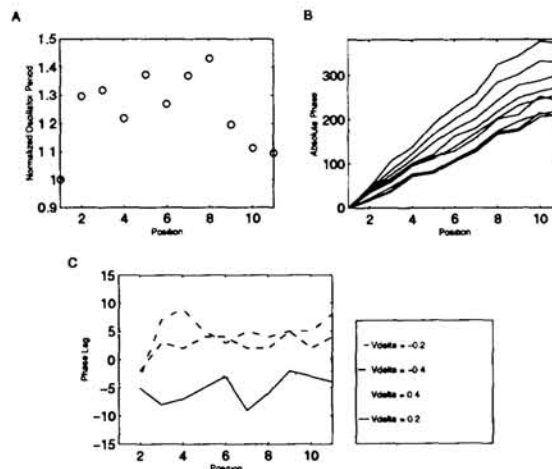

Figure 4: Experimental data obtained from system of coupled oscillators.

quencies.

Another factor that can effect the intersegmental phase lag is the degree of anisotropic coupling. To investigate the effect of asymmetrical coupling, we adjusted $I_{ext}$ in each segment so to produce uniform intrinsic oscillator periods (to within ten percent of 115 ms) along the length of the system. Asymmetrical coupling is established by maintaining $V_{avg} \equiv (V_{ASC} - V_{DES})/2$ at 0.7 volts and varying $V_{delta} \equiv V_{ASC} - V_{DES}$ from 0.4 to - 0.4 volts. $V_{ASC}$ and $V_{DES}$ correspond to the bias voltage that sets the synaptic conductance of presynaptic inputs arriving from the ascending and descending directions respectively. Throughout the experiment, the average of inhibitory (contralateral) and excitatory (ipsilateral) connections from one direction are maintained at equal levels. Figure 4C shows the intersegmental phase lags at different levels of anisotropic coupling. Stronger ascending weights ($V_{delta}$ = 0.4, 0.2 volts) produced negative phase lags, corresponding to backward swimming, while stronger descending connections ($V_{delta}$ = -0.4, -0.2 volts) produce positive phase lags, corresponding to backward swimming. Although mathematical models suggest that stronger ascending coupling should produce forward swimming, we feel that the type of coupling (inhibitory contralateral and excitatory ipsilateral connections) and the oscillatory mode (intrinsic release) of the segmental oscillators may account for this discrepancy.

To study the effects of frequency gradients, we adjusted $I_{ext}$ at each segment such that the that the oscillator period from the head to the tail (from segment 1 to segment 11) varied from 300 ms to 100 ms in 20 ms increments. In addition, to minimize the effect of asymmetrical coupling, we set $V_{avg}$ = 0.8 volts and $V_{delta}$ = 0 volts. The absolute phase under these conditions are shown in Figure 5A. The phase lags are negative with respect to the head position, which corresponds to backward swimming. With a positive frequency gradient, head oscillator at 100 ms and tail oscillator at 300 ms, the resulting phases are in the opposite direction, as shown in Figure 5B. These results are consistent with mathematical models and the *trailing oscillator hypothesis* as expounded by Grillner et. al. (1991).

## 5 CONCLUSIONS AND FUTURE WORK

We have implemented and tested an analog VLSI model of intersegmental coordination with nearest neighbor coupling. We have explored the effects of anisotropic coupling and frequency gradients on system behavior. One of our results—stronger ascending connections produced backward swimming instead of forward swimming—is contrary to theory. There are two factors that may account for this discrepancy: i) our system exhibits inherent spatial disorder in the parameter space due to device mismatch, and ii) the operating point at which we performed the experiments retains high sensitivity to neuron parameter variations and oscillatory modes. We are continuing to explore the parameter space to determine if there are more robust operating points.

We expect that the limitation of our system to only including nearest-neighbor connections is a major factor in the large phase-lag variations that we observed. The importance of both short and long distance connections in the regulation of constant phase under conditions of large variability in the parameter space has been shown by Cohen and Kiemel (1993). To address these issues, we are currently designing a system that facilitates both short and long distance connections (DeWeerth et al, 1997). Additionally, to study the

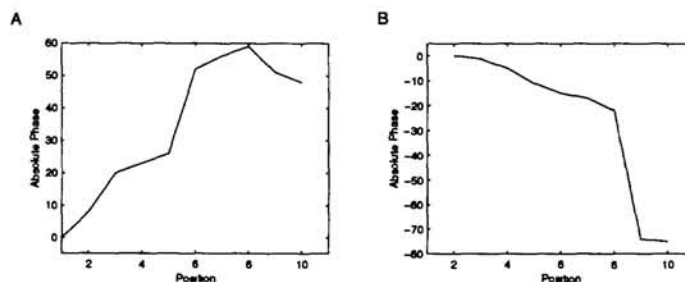

Figure 5: Absolute phase with negative (A) and positive (B) frequency gradients.

role of sensory feedback and to close the loop between neural control and motor behavior, we are also building a mechanical segmented system into which we will incorporate our aVLSI models.

## Acknowledgments

This research is supported by NSF grant IBN-9511721. We would like to thank Avis Cohen for discussion on computational properties that underlie coordinated motor behavior in the lamprey swim system. We would also like to thank Mario Simoni for discussions on pattern generating circuits. We thank the Georgia Tech Analog Consortium for supporting students with travel funds.

## References

Calabrese, R. and De Schutter, E. (1992). Motor-pattern-generating networks in Invertebrates: Modeling Our Way Toward Understanding. *TINS*, 15.11:439–445.

Cohen, A, Holms, P. and Rand R. (1982). The nature of coupling between segmental oscillators of the lamprey spinal generator for locomotion: A mathematical model. J. Math Biol. 13:345–369.

Cohen, A. and Kiemel, T. (1993). Intersegmental coordination: lessons from modeling systems of coupled non-linear oscillators. *Amer. Zool.*, 33:54–65.

DeWeerth, S., Patel, G., Schimmel, D., Simoni, M. and Calabrese, R. (1997). In *Proceedings of the Seventeenth Conference on Advanced Research in VLSI*, R.B. Brown and A.T. Ishii (eds), Los Alamitos, CA: IEEE Computer Society, 182–200.

Friesen, O and Pearce, R. (1993). Mechanisms of intersegmental coordination in leech locomotion. *SINS* 5:41-47.

Getting, P. (1989). A network oscillator underlying swimming in Tritonia. In *Cellular and Neuronal Oscillators*, J.W. Jacklet (ed), New York: Marcel Dekker, 101–128.

Grillner, S. Wallén, P., Brodin, L. and Lansner, A. (1991). Neuronal network generating locomotor behavior in lamprey: circuitry, transmitter, membrane properties, and simulation. *Ann. Rev. Neurosci.*, 14:169–169.

Kopell, N. and Ermentrout, B. (1988). Coupled oscillators and the design of central pattern generators. *Math Biosci.* 90:87–109.

Mahowald, M. and Douglas, R. (1991) A silicon neuron. *Nature*, 354:515–518.

Morris, C. and Lecar, H. (1981) Voltage oscillations in the barnacle giant muscle fiber. *Biophys. J*, 35: 193–213.

Patel, G., DeWeerth, S. (1997). Analogue VLSI Morris-Lecar neuron. *Electronic Letters*, IEE. 33.12: 997-998.

Sigvardt, K.(1993). Intersegmental coordination in the lamprey central pattern generator for locomotion. *SINS* 5:3-15.

Selverston, A. (1989) The Lobster Gastric Mill Oscillator, In *Cellular and Neuronal Oscillators*, J.W. Jacklet (ed), New York: Marcel Dekker, 338–370.

Skinner, F., Kopell, N., and Marder E. (1994) Mechanisms for Oscillation and Frequency Control in Reciprocally Inhibitory Model Neural Networks., *J. of Comp. Neuroscience*, 1:69–87.

Willams, T. (1992). Phase Coupling and Synaptic Spread in Chains of Coupled Neuronal Oscillators. *Science*, vol. 258.

Williams, T., Sigvardt, K. (1994) intersegmental phase lags in the lamprey spinal cord: experimental confirmation of the existence of a boundary region. *J. of Comp. Neuroscience*, 1:61–67.